# Solving Stochastic Games

**Liam Mac Dermed**
College of Computing
Georgia Tech
801 Atlantic Drive
Atlanta, GA 30332-0280
`liam@cc.gatech.edu`

**Charles Isbell**
College of Computing
Georgia Tech
801 Atlantic Drive
Atlanta, GA 30332-0280
`isbell@cc.gatech.edu`

## Abstract

Solving multi-agent reinforcement learning problems has proven difficult because of the lack of tractable algorithms. We provide the first approximation algorithm which solves stochastic games with cheap-talk to within $\epsilon$ absolute error of the optimal game-theoretic solution, in time polynomial in $1/\epsilon$. Our algorithm extends Murray's and Gordon's (2007) modified Bellman equation which determines the *set* of all possible achievable utilities; this provides us a truly general framework for multi-agent learning. Further, we empirically validate our algorithm and find the computational cost to be orders of magnitude less than what the theory predicts.

## 1 Introduction

In reinforcement learning, Bellman's dynamic programming equation is typically viewed as a method for determining the value function — the maximum achievable utility at each state. Instead, we can view the Bellman equation as a method of determining *all* possible achievable utilities. In the single-agent case we care only about the maximum utility, but for multiple agents it is rare to be able to simultaneous maximize all agents' utilities. In this paper we seek to find the *set* of *all* achievable joint utilities (a vector of utilities, one for each player). This set is known as the feasible-set. Given this goal we can reconstruct a proper multi-agent equivalent to the Bellman equation that operates on feasible-sets for each state instead of values.

Murray and Gordon (2007) presented an algorithm for calculating the exact form of the feasible-set based Bellman equation and proved correctness and convergence; however, their algorithm is not guaranteed to converge in a finite number of iterations. Worse, a particular iteration may not be tractable. These are two separate problems. The first problem is caused by the intolerance of an equilibrium to error, and the second results from a potential need for an unbounded number of points to define the closed convex hull that is each states feasible-set. We solve the first problem by targeting $\epsilon$-equilibria instead of exact equilibria, and we solve the second by approximating the hull with a bounded number of points. Importantly, we achieve both solutions while bounding the final error introduced by these approximations. Taken together this produces the first multi-agent reinforcement learning algorithm with theoretical guarantees similar to single-agent value iteration.

## 2 Agenda

We model the world as a fully-observable $n$-player stochastic game with cheap talk (communication between agents that does not affect rewards). Stochastic games (also called Markov games) are the natural multi-agent extension of Markov decision processes with actions being joint actions and rewards being a vector of rewards, one to each player. We assume an implicit inclusion of past joint

actions as part of state (we actually only rely on $\log_2 n + 1$ bits of history containing if and who has defected). We also assume that each player is rational in the game-theoretic sense.

Our goal is to produce a joint policy that is Pareto-optimal (no other viable joint policy gives a player more utility without lowering another player's utility), fair (players agree on the joint policy), and in equilibrium (no player can gain by deviating from the joint policy).[1] This solution concept is the game-theoretic solution.

We present the first approximation algorithm that can efficiently and provably converge to within a given error of game-theoretic solution concepts for all such stochastic games. We factor out the various game theoretic elements of the problem by taking in three functions which compute in turn: the equilibrium $F_{eq}$ (such as correlated equilibrium), the threat $F_{th}$ (such as grim trigger), and the bargaining solution $F_{bs}$ (such as Nash bargaining solution). An error parameters $\epsilon_1$ controls the degree of approximation. The final algorithm takes in a stochastic game, and returns a targeted utility-vector and joint policy such that the policy achieves the targeted utility while guaranteeing that the policy is an $\epsilon_1/(1-\gamma)$-equilibrium (where $\gamma$ is the discount factor) and there are no exact equilibria that Pareto-dominate the targeted utility.

## 3 Previous approaches

Many attempts have been made to extend the Bellman equation to domains with multiple agents. Most of these attempts have focused on retaining the idea of a value function as the memoized solution to subproblems in Bellman's dynamic programming approach (Greenwald & Hall, 2003), (Littman, 2001), (Littman, 2005). This has lead to a few successes particularly in the zero-sum case where the same guarantees as standard reinforcement learning have been achieved (Littman, 2001). Unfortunately, more general convergence results have not been achieved. Recently a negative result has shown that any value function based approach cannot solve the general multi-agent scenario (Littman, 2005). Consider a simple game (Figure 1-A):

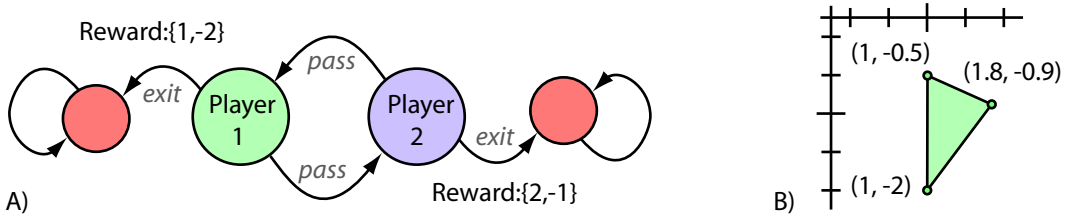

Figure 1: A) The Breakup Game demonstrates the limitation of traditional value-function based approaches. Circles represent states, outgoing arrows represent deterministic actions. Unspecified rewards are zero. B) The final feasible-set for player 1's state ($\gamma = 0.9$).

This game has four states with two terminal states. In the two middle states play alternates between the two players until one of the players decides to exit the game. In this game the only equilibria are stochastic (*E.G.* the randomized policy of each player passing and exiting with probability $\frac{1}{2}$). In each state only one of the agents takes an action, so an algorithm that depends only on a value function will myopically choose to deterministically take the best action, and never converge to the stochastic equilibrium. This result exposed the inadequacy of value functions to capture cyclic equilibrium (where the equilibrium policy may revisit a state).

Several other complaints have been leveled against the motivation behind MAL research following the Bellman heritage. One such complaint is that value function based algorithms inherently target only stage-game equilibria and not full-game equilibria potentially ignoring much better solutions (Shoham & Grenager, 2006). Our approach solves this problem and allows a full-game equilibrium to be reached. Another complaint goes even further, challenging the desire to even target equilibria (Shoham et al., 2003). Game theorists have shown us that equilibrium solutions are correct when agents are rational (infinitely intelligent), so the argument against targeting equilibria boils down to either assuming other agents are not infinitely intelligent (which is reasonable) or that finding

equilibria is not computationally tractable (which we tackle here). We believe that although MAL is primarily concerned with the case when agents are not fully rational, first assuming agents are rational and subsequently relaxing this assumption will prove to be an effective approach.

Murray and Gordon (2007) presented the first multidimensional extension to the Bellman equation which overcame many of the problems mentioned above. In their later technical report (Murray & Gordon, June 2007) they provided an exact solution equivalent to our solution targeting subgame perfect correlated equilibrium with credible threats, while using the Nash bargaining solution for equilibrium selection. In the same technical report they present an approximation method for their exact algorithm that involved sampling the feasible-set. Their approach was a significant step forward; however, their approximation algorithm has no finite time convergence guarantees, and can result in unbounded error.

## 4   Exact feasible-set solution

They key idea needed to extend reinforcement learning into multi-agent domains is to replace the value-function, $V(s)$, in Bellman's dynamic program with a feasible-set function – a mapping from state to feasible-set. As a group of $n$ agents follow a joint-policy, each player $i$ receives rewards. the discounted sum of these rewards is that player's utility, $u_i$. The $n$-dimensional vector $\vec{u}$ containing these utility is known as the joint-utility. Thus a joint-policy yields a joint-utility which is a point in $n$-dimensional space. If we examine *all* (including stochastic) joint-policies starting from state $s$, discard those not in equilibrium, and compute the remaining joint-utilities we will have a set of $n$-dimensional points - the feasible-set. This set is closed and convex, and can be thought of as an $n$-dimensional convex polytope. As this set contains all possible joint-utilities, it will contain the optimal joint-utility for *any* definition of optimal (the bargaining solution $F_{bs}$ will select the utility vector it deems optimal). After an optimal joint-utility has been chosen, a joint-policy can be constructed to achieve that joint-utility using the computed feasible-sets (Murray & Gordon, June 2007). Recall that agents care only about the utility they achieve and not the specific policy used. Thus computing the feasible-set function solves stochastic games, just as computing the value function solves MDPs.

Figure 1-B shows a final feasible-set in the breakup game. The set is a closed convex hull with extreme points $(1, -0.5)$, $(1, -2)$, and $(1.8, -0.9)$. This feasible-set depicts the fact that when starting in player 1's state any full game equilibria will result in a joint-utility that is some weighted average of these three points. For example the players can achieve $(1, -0.5)$ by having player 1 always pass and player 2 exit with probability 0.55. If player 2 tries to cheat by passing when they are supposed to exit, player 1 will immediate exit in retaliation (recall that history is implicitly included in state).

An exact dynamic programing solution falls out naturally after replacing the value-function in Bellman's dynamic program with a feasible-set function, however the changes in variable dimension complicate the backup. An illustration of the modified backup is shown in Figure 2, where steps A-C solve for the action-feasible-set ($Q(s, \vec{a})$), and steps D-E solve for $V(s)$ given $Q(s, \vec{a})$. What is not depicted in Figure 2 is the process of eliminating non-equilibrium policies in steps D-E. We assume an equilibrium filter function $F_{eq}$ is provided to the algorithm, which is applied to eliminate non-equilibrium policies. Details of this process is given in section 5.4. The final dynamic program starts by initializing each feasible-set to be some large over-estimate (a hypercube of the maximum and minimum utilities possible for each player). Each iteration of the backup then contracts the feasible-sets, eliminating unachievable utility-vectors. Eventually the algorithm converges and only achievable joint-utilities remain. The invariant of feasible-sets always overestimating is crucial for guaranteeing correctness, and is a point of great concern below. A more detailed examination of the exact algorithm including a formal treatment of the backup, various game theoretic issues, and convergence proofs are given in Murray and Gordon's technical report (June 2007). This paper does not focus on the exact solution, instead focusing on creating a tractable generalized version.

## 5   Making a tractable algorithm

There are a few serious computational bottlenecks in the exact algorithm. The first problem is that the size of the game itself is exponential in the number of agents because joint actions are

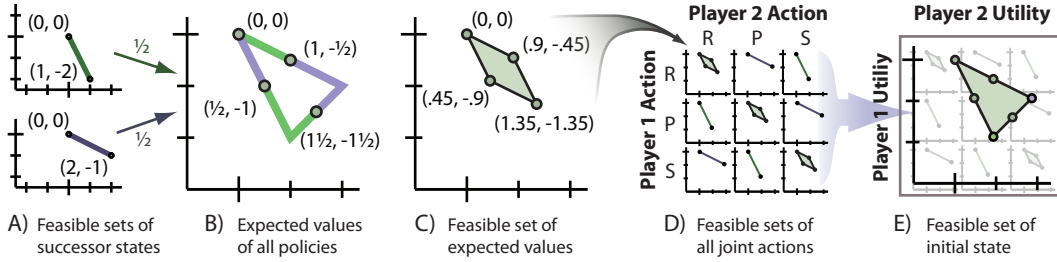

A) Feasible sets of successor states  B) Expected values of all policies  C) Feasible set of expected values  D) Feasible sets of all joint actions  E) Feasible set of initial state

Figure 2: An example of the backup step (one iteration of our modified Bellman equation). The state shown being calculated is an initial rock-paper-scissors game played to decide who goes first in the breakup game from Figure 1. A tie results in a random winner. The backup shown depicts the 2nd iteration of the dynamic program when feasible-sets are initialized to (0,0) and binding contracts are allowed ($F_{eq}$ = set union). In step A the feasibility set of the two successor states are shown graphically. For each combination of points from each successor state the expected value is found (in this case 1/2 of the bottom and 1/2 of the top). These points are shown in step B as circles. Next in step C, the minimum encircling polygon is found. This feasibility region is then scaled by the discount factor and translated by the immediate reward. This is the feasibility-set of a particular joint action from our original state. The process is repeated for each joint action in step D. Finally, in step E, the feasible outcomes of all joint actions are fed into $F_{eq}$ to yield the updated feasibility set of our state.

exponential in the number of players. This problem is unavoidable unless we approximate the game which is outside the scope of this paper. The second problem is that although the exact algorithm always converges, it is not guaranteed to converge in finite time (during the equilibrium backup, an arbitrarily small update can lead to a drastically large change in the resulting contracted set). A third big problem is that maintaining an exact representation of a feasible-set becomes unwieldy (the number of faces of the polytope my blow up, such as if it is curved).

Two important modifications to the exact algorithm allow us to make the algorithm tractable: Approximating the feasible-sets with a bounded number of vertices, and adding a stopping criterion. Our approach is to approximate the feasible-set at the end of each iteration after first calculating it exactly. The degree of approximation is captured by a user-specified parameters: $\epsilon_1$. The approximation scheme yields a solution that is an $\epsilon_1/(1-\gamma)$-equilibrium of the full game while guaranteeing there exists no exact equilibrium that Pareto-dominates the solution's utility. This means that despite not being able to calculate the true utilities at each stage game, if other players did know the true utilities they would gain no more than $\epsilon_1/(1 - \gamma)$ by defecting. Moreover our approximate solution is as good or better than any true equilibrium. By targeting an $\epsilon_1/(1 - \gamma)$-equilibrium we do not mean that the backup's equilibrium filter function $F_{eq}$ is an $\epsilon$-equilibrium (it could be, although making it such would do nothing to alleviate the convergence problem). Instead we apply the standard filter function but stop if no feasible-set has changed by more than $\epsilon_1$.

## 5.1 Consequences of a stopping criterion

Recall we have added a criterion to stop when all feasible-sets contract by less than $\epsilon_1$ (in terms of Hausdorff distance). This is added to ensure that the algorithm makes $\epsilon_1$ absolute progress each iteration and thus will take no more than $O(1/\epsilon_1)$ iterations to converge. After our stopping criterion is triggered the total error present in any state is no more than $\epsilon_1/(1 - \gamma)$ (*i.e.* if agents followed a prescribed policy they would find their actual rewards to be no less than $\epsilon_1/(1 - \gamma)$ promised). Therefore the feasible-sets must represent at least an $\epsilon_1/(1 - \gamma)$-equilibrium. In other words, after a backup each feasible-set is in equilibrium (according to the filter function) with respect to the previous iteration's estimation. If that previous estimation is off by at most $\epsilon_1/(1 - \gamma)$ than the most any one player could gain by deviating is $\epsilon_1/(1 - \gamma)$. Because we are only checking for a stopping condition, and not explicitly targeting the $\epsilon_1/(1 - \gamma)$-equilibrium in the backup we can't guarantee that the algorithm will terminate with the best $\epsilon_1/(1-\gamma)$-equilibrium. Instead we can guarantee that when we do terminate we know that our feasible-sets contain all equilibrium satisfying our original equilibrium filter and no equilibrium with incentive greater than an $\epsilon_1/(1 - \gamma)$ to deviate.

## 5.2 Bounding the number of vertices

Bounding the number of points defining each feasible-set is crucial for achieving a tractable algorithm. At the end of each iteration we can replace each state feasible-set ($V(s)$) with an N point approximation. The computational geometry literature is rich with techniques for approximating convex hulls. However, we want to insure that our feasible estimation is always an over estimation and not an under estimation, otherwise the equilibrium contraction step may erroneously eliminate valid policies. Also, we need the technique to work in arbitrary dimensions and guarantee a bounded number of vertices for a given error bound. A number of recent algorithms meet these conditions and provide efficient running times and optimal worse-case performance (Lopez & Reisner, 2002), (Chan, 2003), (Clarkson, 1993).

Despite the nice theoretical performance and error guarantees of these algorithms they admit a potential problem. The approximation step is controlled by a parameter $\epsilon_2(0 < \epsilon_2 < \epsilon_1)$ determining the maximum tolerated error induced by the approximation. This error results in an expansion of the feasible-set by at most $\epsilon_2$. On the other hand by targeting $\epsilon_1$-equilibrium we can terminate if the backups fail to make $\epsilon_1$ progress. Unfortunately this $\epsilon_1$ progress is not uniform and may not affect much of the feasible-set. If this is the case, the approximation expansion could potentially expand past the original feasible-set (thus violating our need for progress to be made every iteration, see Figure 3-A). Essentially our approximation scheme must also insure that it is a subset of the previous step's approximation. With this additional constraint in mind we develop the following approximation inspired by (Chen, 2005):

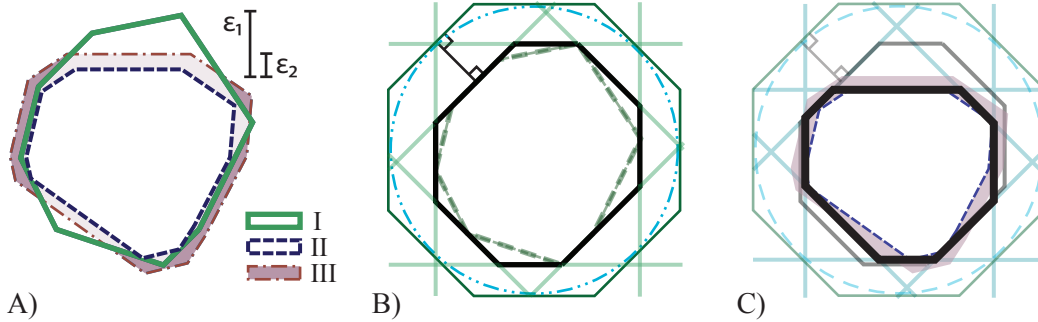

Figure 3: A) (I) Feasible hull from previous iteration. (II) Feasible hull after equilibrium contraction. The set contracts at least $\epsilon_1$. (III) Feasible hull after a poor approximation scheme. The set expands at most $\epsilon_2$, but might sabotage progress. B) The hull from A-I is approximated using halfspaces from a given regular approximation of a Euclidean ball. C) Subsequent approximations using the same set of halfspaces will not backtrack.

We take a fixed set of hyperplanes which form a regular approximation of a Euclidean ball such that the hyperplane's normals form an angle of at most $\theta$ with their neighbors (*E.G.* an optimal Delaunay triangulation). We then project these halfspaces onto the polytope we wish to approximate (*i.e.* retain each hyperplanes' normals but reduce their offsets until they touch the given polytope). After removing redundant hyperplanes the resulting polytope is returned as the approximation (Figure 3-B). To insure a maximum error of $\epsilon_2$ with $n$ players: $\theta \leq 2 \arccos[(r/(\epsilon_2 + r))^{1/n}]$ where $r = R_{max}/(1 - \gamma)$.

The scheme trivially uses a bounded number of facets (only those from the predetermined set), and hence a bounded number of vertices. Finally, by using a fixed set of approximating hyperplanes successive approximations will strictly be subsets of each other - no hyperplane will move farther away when the set its projecting onto shrinks (Figure 3-C). After both the $\epsilon_1$-equilibrium contraction step and the $\epsilon_2$ approximation step we can guarantee at least $\epsilon_1 - \epsilon_2$ progress is made. Although the final error depends only on $\epsilon_1$ and not $\epsilon_2$, the rate of convergence and the speed of each iteration is heavily influenced by $\epsilon_2$. Our experiments (section 6) suggest that the theoretical requirement of $\epsilon_2 < \epsilon_1$ is far too conservative.

## 5.3 Computing expected feasible-sets

Another difficulty occurs during the backup of $Q(s, \vec{a})$. Finding the expectation over feasible-sets involves a modified set sum (step B in fig 2), which naively requires an exponential looping over all possible combinations of taking one point from the feasible-set of each successor state. We can help the problem by applying the set sum on an initial two sets and fold subsequent sets into the result. This leads to polynomial performance, but to an uncomfortably high-degree. Instead we can describe the problem as the following multiobjective linear program (MOLP):

> **Simultaneously maximize** foreach player $i$ from 1 to $n$: $\sum_{s'} \sum_{\vec{v} \in V(s')} v_i x_{s'\vec{v}}$
> **Subject to:** for every state $s'$ $\sum_{\vec{v} \in V(s')} x_{s'\vec{v}} = P(s'|s, \vec{a})$

where we maximize over variables $x_{s'\vec{v}}$ (one for each $\vec{v} \in V(s')$ for all $s'$) and $\vec{v}$ is a vertex in the feasible-set $V(s')$ and $v_i$ is the value of that vertex to player $i$. This returns only the Pareto frontier. An optimized version of the algorithm described in this paper would only need the frontier, not the full set as calculating the frontier depends only on the frontier (unless the threat function needs the entire set). For the full feasible-set $2^n$ such MOLPs are needed, one for each orthant.

Like our modified view of the Bellman equation as trying to find the entire set of achievable policy payoffs so too can we view linear programming as trying to find the entire set of achievable values of the objective function. When there is a single objective function this is simply a maximum and minimum value. When there is more than one objective function the solution then becomes a multidimensional convex set of achievable vectors. This problem is known as multiobjective linear programming and has been previously studied by a small community of operation researchers under the umbrella subject of multiobjective optimization (Branke et al., 2005). MOLP is formally defined as a technique to find the Pareto frontier of a set of linear objective functions subject to linear inequality constraints. The most prominent exact method for MOLP is the Evans-Steuer algorithm (Branke et al., 2005).

## 5.4 Computing correlated equilibria of sets

Our generalized algorithm requires an equilibrium-filter function $F_{eq}$. Formally this is a monotonic function $F_{eq} : \mathcal{P}(R^n) \times \ldots \times \mathcal{P}(R^n)) \rightarrow \mathcal{P}(R^n)$ which outputs a closed convex subset of the smallest convex set containing the union of the input sets. Here $\mathcal{P}$ denotes the powerset. It is monotonic as $x \subseteq y \Rightarrow F_{eq}(x) \subseteq F_{eq}(y)$. The threat function $F_{th}$ is also passed to $F_{eq}$. Note than requiring $F_{eq}$ to return a closed convex set disqualifies Nash equilibria and its refinements. Due to the availability of cheap talk, reasonable choices for $F_{eq}$ include correlated equilibria (CE), $\epsilon$-CE, or a coalition resistant variant of CE. Filtering non-equilibrium policies takes place when the various action feasible-sets ($Q$) are merged together as shown in step E of Figure 2. Constructing $F_{eq}$ is more complicated than computing the equilibria for a stage game so we describe below how to target CE.

For a normal-form game the set of correlated equilibria can be determined by taking the intersection of a set of halfspaces (linear inequality constraints) (Greenwald & Hall, 2003). Each variable of these halfspaces represents the probability that a particular joint action is chosen (via a shared random variable) and each halfspace represents a rationality constraint that a player being told to take one action would not want to switch to another action. There are $\sum_1^n |A_i|(|A_i| - 1)$ such rationality constraints (where $|A_i|$ is the number of actions player $i$ can take).

Unlike in a normal-form game, the rewards for following the correlation device or defecting (switching actions) are not directly given in our dynamic program. Instead we have a feasible-set of possible outcomes for each joint action $Q(s, \vec{a})$ and a threat function $F_{th}$. Recall that when following a policy to achieve a desired payoff, not only must a joint action be given, but also subsequent payoffs for each successor state. Thus the halfspace variables must not only specify probabilities over joint actions but also the subsequent payoffs (a probability distribution over the extreme points of each successor feasible-set). Luckily, a mixture of probability distributions is still a probability distribution so our final halfspaces now have $\sum_{\vec{a}} |Q(s, \vec{a})|$ variables (we still have the same number of halfspaces with the same meaning as before).

At the end of the day we do not want feasible probabilities over successor states, we want the utility-vectors afforded by them. To achieve this without having to explicitly construct the polytope

described above (which can be exponential in the number of halfspaces) we can describe the problem as the following MOLP (given $Q(s, \vec{a})$ and $F_{th}$):

**Simultaneously maximize** foreach player $i$ from 1 to $n$:  $\sum_{\vec{a}\vec{u}} u_i x_{\vec{a}\vec{u}}$
**Subject to:**  probability constraints $\sum x_{\vec{a}\vec{u}} = 1$ and $x_{\vec{a}\vec{u}} \geq 0$
and foreach player $i$, actions $a_1, a_2 \in A_i$, $(a_2 \neq a_1)$
$\sum_{\vec{a}\vec{u}|a_i=a_1} u_i x_{\vec{a}\vec{u}} \geq \sum_{\vec{a}\vec{u}|a_i=a_2} F_{th}(s, \vec{a}) x_{\vec{a}\vec{u}}$

where variables $x_{\vec{a}\vec{u}}$ represent the probability of choosing joint action $\vec{a}$ and subsequent payoff $\vec{u} \in Q(s, \vec{a})$ in state $s$ and $u_i$ is the utility to player $i$.

## 5.5 Proof of correctness

Murray and Gordon (June 2007) proved correctness and convergence for the exact algorithm by proving four properties: 1) Monotonicity (feasible-sets only shrink), 2) Achievability (after convergence, feasible-sets contain only achievable joint-utilities), 3) Conservative initialization (initialization is an over-estimate), and 4) Conservative backups (backups don't discard valid joint-utilities). We show that our approximation algorithm maintains these properties.

1) Our feasible-set approximation scheme was carefully constructed so that it would not permit backtracking, maintaining monotonicity (all other steps of the backup are exact). 2) We have broadened the definition of achievability to permit $\epsilon_1/(1 - \gamma)$ error. After all feasible-sets shrink by less than $\epsilon_1$ we could modify the game by giving a bonus reward less than $\epsilon_1$ to each player in each state (equal to that state's shrinkage). This modified game would then have converged exactly (and thus would have a perfectly achievable feasible-set as proved by Murray and Gordon). Any joint-policy of the modified game will yield at most $\epsilon_1/(1 - \gamma)$ more than the same joint-policy of our original game thus all utilities of our original game are off by at most $\epsilon_1/(1 - \gamma)$. 3) Conservative initialization is identical to the exact solution (start with a huge hyperrectangle with sides $R^i_{max}/(1 - \gamma)$). 4) Backups remain conservative as our approximation scheme never underestimates (as shown in section 5.2) and our equilibrium filter function $F_{eq}$ is required to be monotonic and thus will never underestimate if operating on overestimates (this is why we require monotonicity of $F_{eq}$). CE over sets as presented in section 5.4 is monotonic. Thus our algorithm maintains the four crucial properties and terminates with all exact equilibria (as per conservative backups) while containing no equilibrium with error greater than $\epsilon_1/(1 - \gamma)$.

## 6 Empirical results

We implemented a version of our algorithm targeting exact correlated equilibrium using grim trigger threats (defection is punished to the maximum degree possible by all other players, even at one's own expense). The grim trigger threat reduces to a 2 person zero sum game where the defector receives their normal reward and all other players receive the opposite reward. Because the other players receive the same reward in this game they can be viewed as a single entity. Zero sum 2-player stochastic games can be quickly solved using FFQ-Learning (Littman, 2001). Note that grim trigger threats can be computed separately before the main algorithm is run. When computing the threats for each joint action, we use the GNU Linear Programming Kit (GLPK) to solve the zero-sum stage games. Within the main algorithm itself we use ADBASE (Steuer, 2006) to solve our various MOLPs. Finally we use QHull (Barber et al., 1995) to compute the convex hull of our feasible-sets and to determine the normals of the set's facets. We use these normals to compute the approximation. To improve performance our implementation does not compute the entire feasible hull, only those points on the Pareto frontier. A final policy will exclusively choose targets from the frontier (using $F_{bs}$) (as will the computed intermediate equilibria) so we lose nothing by ignoring the rest of the feasible-set (unless the threat function requires other sections of the feasible-set, for instance in the case of credible threats). In other words, when computing the Pareto frontier during the backup the algorithm relies on no points except those of the previous step's Pareto frontier. Thus computing only the Pareto frontier at each iteration is not an approximation, but an exact simplification.

We tested our algorithm on a number of problems with known closed form solutions, including the breakup game (Figure 4). We also tested the algorithm on a suite of random games varying across the number of states, number of players, number of actions, number of successor states (stochasticity of

the game), coarseness of approximation, and density of rewards. All rewards were chosen at random between 1 and -1, and $\gamma$ was always set to 0.9.

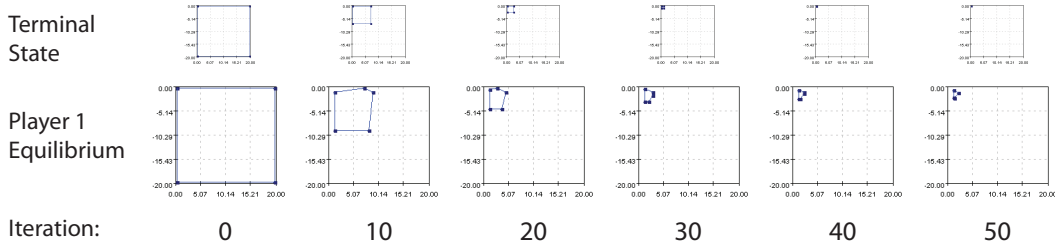

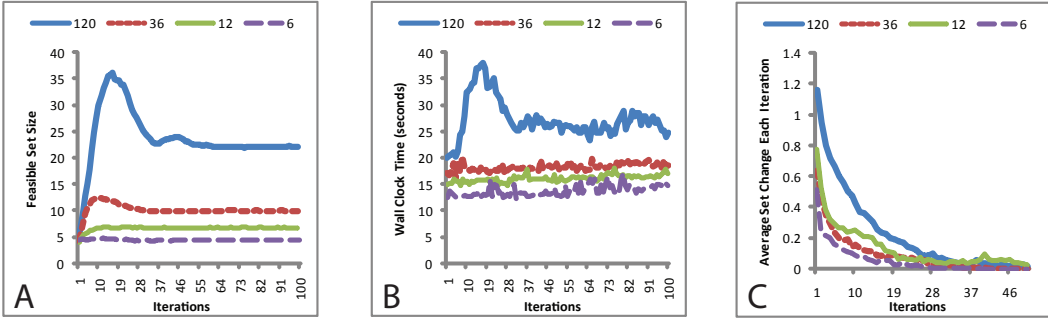

Figure 4: A visualization of feasible-sets for the terminal state and player 1's state of the breakup game at various iterations of the dynamic program. By the 50th iteration the sets have converged.

An important empirical question is what degree of approximation should be adopted. Our testing (see Figure 5) suggests that the theoretical requirement of $\epsilon_2 < \epsilon_1$ is overly conservative. While the bound on $\epsilon_2$ is theoretically proportional to $R_{max}/(1 - \gamma)$ (the worst case scale of the feasible-set) a more practical choice for $\epsilon_2$ would be in scale with the final feasible-sets (as should a choice for $\epsilon_1$).

Figure 5: Statistics from a random game (100 states, 2 players, 2 actions each, with $\epsilon_1 = 0.02$ ) run with different levels of approximation. The numbers shown (120, 36, 12, and 6) represent the number of predetermined hyperplanes used to approximate each Pareto frontier. A) The better approximations only use a fraction of the hyperplanes available to them. B) Wall clock time is directly proportional to the size of the feasible-sets. C) Better approximations converge more each iteration (the coarser approximations have a longer tail), however due to the additional computational costs the 12 hyperplane approximation converged quickest (in total wall time). The 6, 12, and 36 hyperplane approximations are insufficient to guarantee convergence ($\epsilon_2 = 0.7, 0.3, 0.1$ respectively) yet only the 6-face approximation occasionally failed to converge.

## 6.1 Limitations

Our approach is overkill when the feasible-sets are one dimensional (line segments) (as when the game is zero-sum, or agents share a reward function), because CE-Q learning will converge to the correct solution without additional overhead. When there are no cycles in the state-transition graph (or one does not wish to consider cyclic equilibria) traditional game-theory approaches suffice. In more general cases, our algorithm brings significant advantages. However despite scaling linearly with the number of states, the multiobjective linear program for computing the equilibrium hull scales very poorly. The MOLP remains tractable only up to about 15 joint actions (which results in a few hundred variables and a few dozen constraints, depending on feasible-set size). This in turn prevents the algorithm from running with more than four agents.

## Footnotes

[1]The precise meaning of fair, and the type of equilibrium is intentionally left unspecified for generality.

# References

Barber, C. B., Dobkin, D. P., & Huhdanpaa, H. (1995). The quickhull algorithm for convex hulls. *ACM Transactions on Mathematical Software*, *22*, 469–483.

Branke, J., Deb, K., Miettinen, K., & Steuer, R. E. (Eds.). (2005). *Practical approaches to multi-objective optimization, 7.-12. november 2004*, vol. 04461 of *Dagstuhl Seminar Proceedings*. Internationales Begegnungs- und Forschungszentrum (IBFI), Schloss Dagstuhl, Germany IBFI, Schloss Dagstuhl, Germany.

Chan, T. M. (2003). Faster core-set constructions and data stream algorithms in fixed dimensions. *Comput. Geom. Theory Appl* (pp. 152–159).

Chen, L. (2005). New analysis of the sphere covering problems and optimal polytope approximation of convex bodies. *J. Approx. Theory*, *133*, 134–145.

Clarkson, K. L. (1993). Algorithms for polytope covering and approximation, and for approximate closest-point queries.

Greenwald, A., & Hall, K. (2003). Correlated-q learning. *Proceedings of the Twentieth International Conference on Machine Learning* (pp. 242–249).

Littman, M. L. (2001). Friend-or-foe Q-learning in general-sum games. *Proc. 18th International Conf. on Machine Learning* (pp. 322–328). Morgan Kaufmann, San Francisco, CA.

Littman, M. Z. . A. G. . M. L. (2005). Cyclic equilibria in markov games. *Proceedings of Neural Information Processing Systems*. Vancouver, BC, Canada.

Lopez, M. A., & Reisner, S. (2002). Linear time approximation of 3d convex polytopes. *Comput. Geom. Theory Appl.*, *23*, 291–301.

Murray, C., & Gordon, G. (June 2007). *Finding correlated equilibria in general sum stochastic games* (Technical Report). School of Computer Science, Carnegie Mellon University.

Murray, C., & Gordon, G. J. (2007). Multi-robot negotiation: Approximating the set of subgame perfect equilibria in general-sum stochastic games. In B. Schölkopf, J. Platt and T. Hoffman (Eds.), *Advances in neural information processing systems 19*, 1001–1008. Cambridge, MA: MIT Press.

Shoham, Yoav, P., & Grenager (2006). If multi-agent learning is the answer, what is the question? *Artificial Intelligence*.

Shoham, Y., Powers, R., & Grenager, T. (2003). *Multi-agent reinforcement learning: a critical survey* (Technical Report).

Steuer, R. E. (2006). Adbase: A multiple objective linear programming solver for efficient extreme points and unbounded efficient edges.

